# Self Organizing Neural Networks for the Identification Problem

Manoel Fernando Tenorio

School of Electrical Engineering

Purdue University

W. Lafayette, IN. 47907

tenorio@ee.ecn.purdue.edu

Wei-Tsih Lee

School of Electrical Engineering

Purdue University

W. Lafayette, IN. 47907

lwt@ed.ecn.purdue.edu

## ABSTRACT

This work introduces a new method called Self Organizing Neural Network (SONN) algorithm and demonstrates its use in a system identification task. The algorithm constructs the network, chooses the neuron functions, and adjusts the weights. It is compared to the Back-Propagation algorithm in the identification of the chaotic time series. The results shows that SONN constructs a simpler, more accurate model, requiring less training data and epochs. The algorithm can be applied and generalized to appilications as a classifier.

## I. INTRODUCTION

### I.1 THE SYSTEM IDENTIFICATION PROBLEM

In various engineering applications, it is important to be able to estimate, interpolate, and extrapolate the behavior of an unknown system when only its input-output pairs are available. Algorithms which produce an estimation of the system behavior based on these pairs fall under the category of system identification techniques.

### I.2 SYSTEM IDENTIFICATION USING NEURAL NETWORKS

A general form to represent systems, both linear and nonlinear, is the Kolmogorov-Garbor polynomial [Garbor, 1961 ] shown below:

$$y = a_0 + \sum_i a_i x_i + \sum_i \sum_j a_{ij} x_i x_j + \cdots$$

(1)

where the y is the output, and x the input to the system. [Garbor ,1961] proposed a learning method that adjusted the coefficient of (1) by minimizing the mean square error between each desired output sample and the actual output.

This paper describes a supervised learning algorithm for structure construction and adjustment. Here, systems which can be described by (1) are presented. The computation of the function for each neuron performs a choice from a set of possible functions previously assigned to the algorithm, and it is general enough to accept a wide range of both continuous and discrete functions. In this work, the set is taken from variants of the 2-input quadratic polynomial for simplicity, although there is no requirement making it so. This approach abandons the simplistic mean-square error for performance measure in favor of a modified Minimum Description Length (MDL) criterion [Rissanen,1975], with provisions to measure the complexity of the model generated. The algorithm searches for the simplest model which generates the best estimate. The modified MDL, from hereon named the Structure Estimation Criterion (SEC), is applied hierarchically in the selection of the optimal neuron transfer function from the function set, and then used as an optimal criterion to guide the construction of the structure. The connectivity of the resulting structure is arbitrary, and under the correct conditions [Geman&Geman, 84] the estimation of the structure is optimal in terms of the output error and low function complexity.This approach shares the same spirit of GMDH-type algorithms. However, the concept of parameter estimation from Information Theory, combined with a stochastic search algorithm - Simulated Annealing, was used to create a new tool for system identification.

This work is organized as follows: section II presents the problem formulation and the Self Organizing Neural Network (SONN) algorithm description; section III describes the results of the application of SONN to a well known problem tested before using other neural network algorithms [Lapedes&Farber, 1987; Moody, 1988]; and finally, section IV presents a discussion of the results and future directions for this work.

## II. THE SELF ORGANIZING NEURAL NETWORK ALGORITHM

## II.1 SELF ORGANIZING STRUCTURES

The Self Organizing Neural Network (SONN) algorithm performs a search on the model space by the construction of hypersurfaces. A network of nodes, each node representing a hypersurface, is organized to be an approximate model of the real system. SONN can be fully characterized by three major components, which can be modified to incorporate knowledge about the process: (1) a generating rule of the primitive neuron transfer functions, (2) an evaluation method which accesses the quality of the model, and, (3) a structure search strategy. Below, the components of SONN are discussed.

## II.2 THE ALGORITHM STRUCTURE

## II.2.1 The Generating Rule

Given a set of observations S:

$$S = \{(X_1, Y_1),(X_2, Y_2),\cdots,(X_l, Y_l)\} \qquad \text{generated by}$$

$$Y_i = f(X_i) + \eta \qquad (2)$$

where f(.) is represented by a Kolmogorov-Garbor polynomial, and the random variable $\eta$ is normally distributed, $N(0,1)$. The dimensions of Y is m, and the dimensions of X is n. Every component $y_k$ of Y forms a hypersurface $y_k = f_k(X)$ in the space of *dim* (X) + 1. The problem is to find f(.), given the observations S, which is a corrupted version of the desired function. In this work, the model which estimates f(.) is desired to be as accurate and simple (small number of parameters, and low degree of non linearity) as possible.

The approach taken here is to estimate the simplest model which best describes f(.) by generating optimal functions for each neuron, which can be viewed as the construction of a hypersurface based on the observed data. It can be described as follows: given a set of observations S; use p components of the n dimensional space of X to create a hypersurface which best describes $y_k = f(X)$, through a three step process. First, given $X = [x_1, x_2, x_3, ..., x_n]$ and $y_k$, and the mapping $\Psi_n$: $[x_1, x_2, x_3, ..., x_n] \rightarrow [x_{\Psi(1)}, x_{\Psi(2)}, x_{\Psi(3)}, ..., x_{\Psi(n)}]$, construct the hypersurface $h_1(x_{\Psi(1)}, x_{\Psi(2)}, x_{\Psi(3)}, ..., x_{\Psi(n)})$ ($h_i$ after the first iteration) of p+1 dimensions, where $\Psi_n$ is a projection from n dimensions to p dimensions. The elements of the domain of $\Psi_n$ are called terminals. Second, If the global optimality criterion is reached by the construction of $h_i(x_{\Psi(1)}, x_{\Psi(2)}, x_{\Psi(3)}, ..., x_{\Psi(n)})$, then stop, otherwise continue to the third step. Third, generate from $[x_1, x_2, x_3, ..., x_n, h_1(x_{\Psi(1)}, x_{\Psi(2)}, x_{\Psi(3)}, ..., x_{\Psi(n)})]$ a new p+1 dimensional hypersurface $h_{i+1}$ through the extended mapping $\Psi_{n+1}(.)$, and reapply the second step. The resulting model is a multilayered neural network whose topology is arbitrarily complex and created by a stochastic search guided by a structure estimation criterion. For simplicity in this work, the set of prototype functions (F) is restricted to be 2-input quadratic surfaces or smaller, with only four possible types:

$$y = a_0 + a_1 x_1 + a_2 x_2 \qquad (3)$$

$$y = a_0 + a_1 x_1 + a_2 x_2 + a_3 x_1 x_2 \qquad (4)$$

$$y = a_0 + a_1 x_1 + a_2 x_1^2 \qquad (5)$$

$$y = a_0 + a_1 x_1 + a_2 x_2 + a_3 x_1 x_2 + a_4 x_1^2 + a_5 x_2^2 \qquad (6)$$

## II.2.2 Evaluation of the Model Based on the MDL Criterion

The selection rule (T) of the neuron transfer function was based on a modification of the Minimal Description Length (MDL) information criterion. In [Rissanen, 1975] the principle of minimal description for statistical estimation was developed. The MDL provides a trade-off between the accuracy and the complexity of the model by including the structure estimation term of the final model. The final model (with the minimal

MDL) is optimum in the sense of being a consistent estimate of the number of parameters while achieving the minimum error [Rissanen,1980]. Given a sequence of observation $x_1,x_2,x_3,...,x_N$ from the random variable X, the dominant term of the MDL in [Rissanen, 1975] is:

$$MDL = - \log f(x|\theta) + 0.5 \ k \log N \qquad (7)$$

where $f(x|\theta)$ is the estimated probability density function of the model, k is the number of parameters, and N is the number of observations. The first term is actually the negative of the maximum likelihood (ML) with respect to the estimated parameter. The second term describes the structure of the models and it is used as a penalty for the complexity of the model. In the case of linear polynomial regression, the MDL is:

$$MDL = - 0.5 \ N \log S_n^2 + 0.5 \ k \log N \qquad (8)$$

where k is the number of coefficients in the model selected.

In the SONN algorithm, the MDL criterion is modified to operate both recursively and hierarchically. First, the concept of the MDL is applied to each candidate prototype surface for a given neuron. Second, the acceptance of the node, based on Simulated Annealing, uses the MDL measure as the system energy. However, since the new neuron is generated from terminals which can be the output of other neurons, the original definition of the MDL is unable to compute the true number of system parameters of the final function. Recall that due to the arbitrary connectivity, feedback loops and other configurations it is non trivial to compute the number of parameters in the entire structure. In order to reflect the hierarchical nature of the model, a modified MDL called Structure Estimation Criterion (SEC) is used in conjunction with an heuristic estimator of the number of parameters in the system at each stage of the algorithm. A computationally efficient heuristic for the estimation of the number of parameters in the model is based on the fact that SONN creates a tree-like structure with multiple roots at the input terminals. Then k, in expression (8), can be estimated recursively by:

$$k = k_L + k_R + (\text{no. of parameters of the current node}) \qquad (9)$$

where $k_L$ and $k_R$ are the estimated number of parameters of the left and right parents of the current node, respectively. This heuristic estimator is neither a lower bound nor an upper bound of the true number of parameter in the model.

### II.2.3 The SONN Algorithm
To explain the algorithm, the following definitions are necessary: Node - neuron and the associated function, connections, and SEC; BASIC NODE - A node for the system input variable; FRONT NODE - A node without children; INTERMIDIATE NODE - The nodes that are neither front or basic nodes; STATE - The collection of nodes, and the configuration of their interconnection; INITIAL STATE $(S_I)$ - The state with only basic nodes; PARENT AND CHILD STATE - The child state is equal to the parent state except for f a new node and its interconnection generated on the parent state structure; NEIGHBOR STATE - A state that is either a child or a parent state of another; ENERGY

OF THE STATE (SEC-$S_i$) - The energy of the state is defined as the minimum SEC of all the front nodes in that state.

In the SONN algorithm, the search for the correct model structure is done via Simulated Annealing. Therefore the algorithm at times can accept partial structures that look less than ideal. In the same way, it is able to discard partially constructed substructures in search for better results. The use of this algorithm implies that the node accepting rule (R) varies at run-time according to a cooling temperature (T) schedule. The SONN algorithm is as follows:

*Initialize T, and $S_I$*

*Repeat*

> *Repeat*
>> $Sj = generate\ (Si),$                                            - application of P.
>> *If accept ( SEC_Sj, SEC_Si, T) then Si = Sj,*      - application of R.
> *until the number of new neurons is greater than N.*
> *Decrease the temperature T.*

*until The temperature T is smaller than $T_{end}$ (Terminal temperature for Simulated Annealing).*

Each neuron output and the system input variables are called terminals. Terminals are viewed as potential dimensions from which a new hypersurface can be constructed. Every terminal represents the best tentative to approximate the system function with the available information, and are therefore treated equally.

## III. EXAMPLE - THE CHAOTIC TIME SERIES

In the following results, the chaotic time series generated by the Mackay-Glass differential equations was used. The SONN with the SEC, and its heuristic variant were used to obtain the approximate model of the system. The result is compared with those obtained by using the nonlinear signal processing method [Lapedes&Farber, 1987] . The advantages and disadvantages of both approaches are analyzed in the next section.

### III.1 Structure of the Problem

The MacKay-Glass differential equation used here can be described as:

$$\frac{\partial x(t)}{\partial t} = \frac{a\,x(t - \tau)}{1 + x^{10}(t - \tau)} - b\,x(t)$$

$$(10)$$

By setting $a = 0.2$, $b = 0.1$, and $\tau = 17$, a chaotic time series with a`strange attractor of fractal dimension about 3.5 will be produced [Lapedes&Farber, 1987] . To compare the accuracy of prediction the normalized root mean square error is used as a performance index:

$$\text{normalized RMSE} = \frac{\text{RMSE}}{\text{Standard Deviation}} \qquad (11)$$

## III.2. SONN WITH THE HEURISTIC SEC (SONN-H)

In the following examples, a modified heuristic version of the SEC is used. The estimator of the number of parameters is given by (9), and the final configuraion is shown in figure 1.

### III.2.1 Node 19

In this subsection, SONN is allowed to generate up to the 19th accepted node. In this first version of the algorithm, all neurons have the same number of interconnections, and therefore draw their transfer function from the same pool of functions. . Generalizations of the algorithm can be easily made to accommodate multiple input functions, and neuron transfer function assignment being drawn from separate pools. In this example, the first one hundred points of the time series was used for training, and samples 101 through 400 used for prediction testing. The total number of weights in the network is 27. The performance index average 0.07. The output of the network is overlapped in the figure 2 with the original time series.

For comparison purposes, a GDR network with the structure used in [Lapedes&Farber, 1987] is trained for 6500 epochs. The training data consisted of the first 500 points of the time series, and the testing data ran from the 501st sample to the 832nd. The total number of weights is 165, and the final performance index equal to 0.12. This was done to give both algorithms similar computational resources. Figure 3 shows the original time series overlapped with the GDR network output.

### III.2.2 NODE 37

In this subsection, the model chosen was formed by the $37^{th}$ accepted node. The network was trained in a similar manner to the first example, since it is part of the same run. The final number of weights is 40, and the performance index 0.018. Figure 4 shows the output of the network overlapped with the original time series. Figure 5 shows the GDR with 11,500 epochs. Notice that in both cases, the GDR network demands 150 connections and 150 weights, as compared to 12 connections and 27 weights for the first example and 10 connections and 40 weights for the second example. The comparison of the performance of different models is listed in figure 6.

## IV. Conclusion and Future Work

In this study, we proposed a new approach for the identification problem based on a flexible, self-organizing neural network (SONN) structure. The variable structure provides the opportunity to search and construct the optimal model based on input-output observations. The hierarchical version of the MDL, called the structure estimation criteria,

was used to guide the trade-off between the model complexity and the accuracy of the estimation. The SONN approach demonstrates potential usefulness as a tool for system identification through the example of modeling a chaotic time series.

## REFERENCE

Garber, D., et. al. ,"A universal nonlinear filter, predicator and simulator which optimizes itsekf by a learning process," IEE Proc.,18B, pp. 422-438, 1961

Rissanen,J. "Modeling by shortest data description," Automatica, vol.14, pp. 465-471,1975

Gemen, S, and Gemen D., "Stochastic relaxation, gibbs deisribution, and the bayesian restoration of images," IEEE PAMI., PAMI-6,pp.721-741, 1984

Lapedes,A. and Farber, R. ,"Nonlinear signal processing using neural networks: Predication and system modeling," TR. LA-UR-87-2662, 1987

Moody, J.   This volume

Rissanen ,J. "Consistent order estimation of autoregressive processing by shortest description of data," Analysis and optimization of stochastic system, Jacobs et. al. Eds. N.Y. Academic, 1980

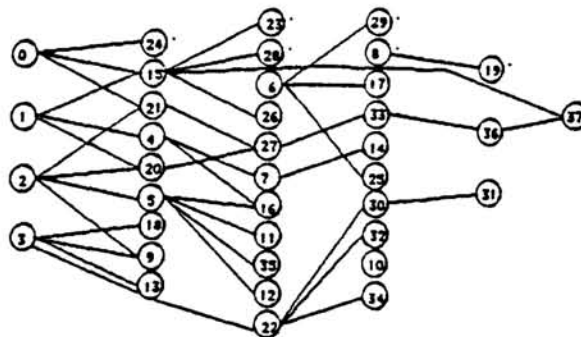

**Figure 1. The 37th State Generated**

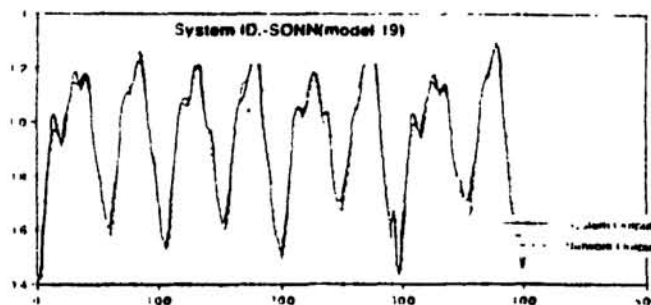

**Figure 2. SONN 19th Model, P.I. = 0.06**

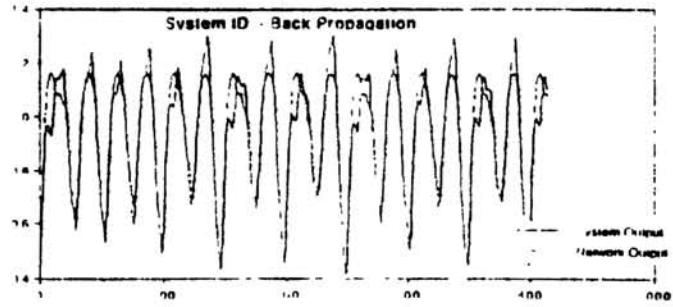

**Figure 3.** GDR after 6,500 Epochs, P.I. = 0.12

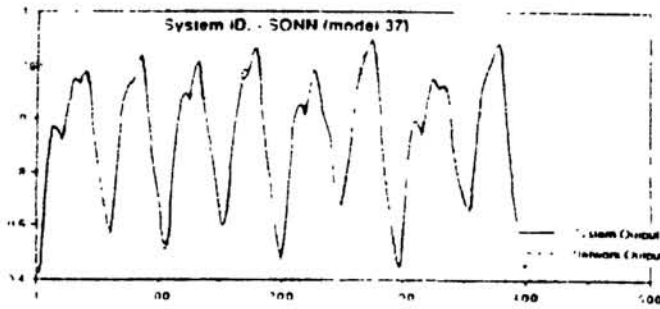

**Figure 4.** SONN 37th Model, P.I. = 0.038

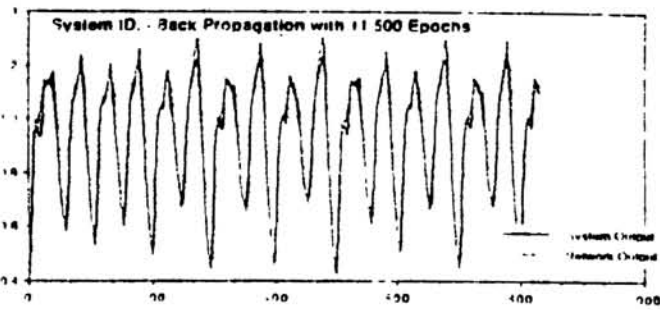

**Figure 5.** GDR after 11,500 Epochs, P.I. = 0.018

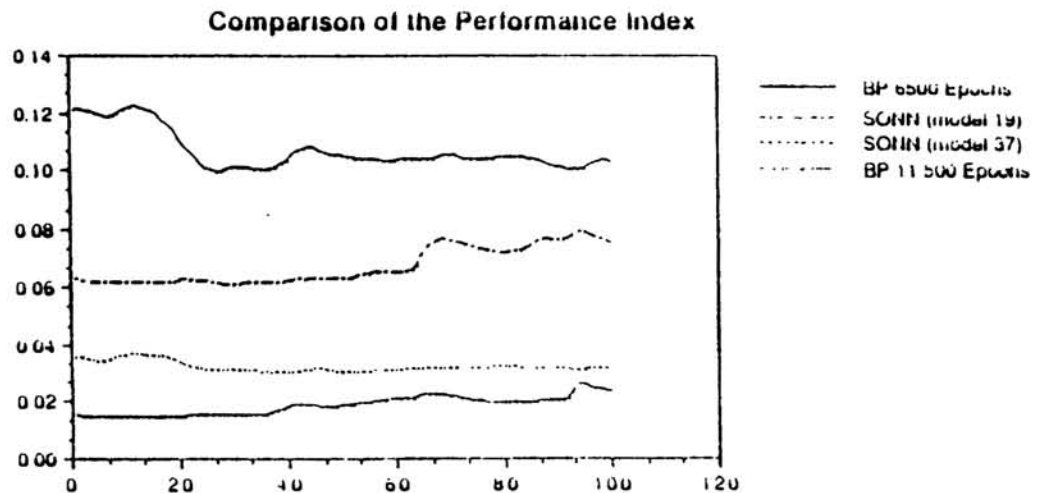

**Figure 6.** Performance Index Versus the Number of Predicted Points